# Recurrent Eye Tracking Network Using a Distributed Representation of Image Motion

**P. A. Viola**
Artificial Intelligence Laboratory
Massachusetts Institute of Technology

**S. G. Lisberger**
Department of Physiology
W.M. Keck Foundation Center for Integrative Neuroscience
Neuroscience Graduate Program
University of California, San Francisco

**T. J. Sejnowski**
Salk Institute, Howard Hughes Medical Institute
Department of Biology
University of California, San Diego

## Abstract

We have constructed a recurrent network that stabilizes images of a moving object on the retina of a simulated eye. The structure of the network was motivated by the organization of the primate visual target tracking system. The basic components of a complete target tracking system were simulated, including visual processing, sensory-motor interface, and motor control. Our model is simpler in structure, function and performance than the primate system, but many of the complexities inherent in a complete system are present.

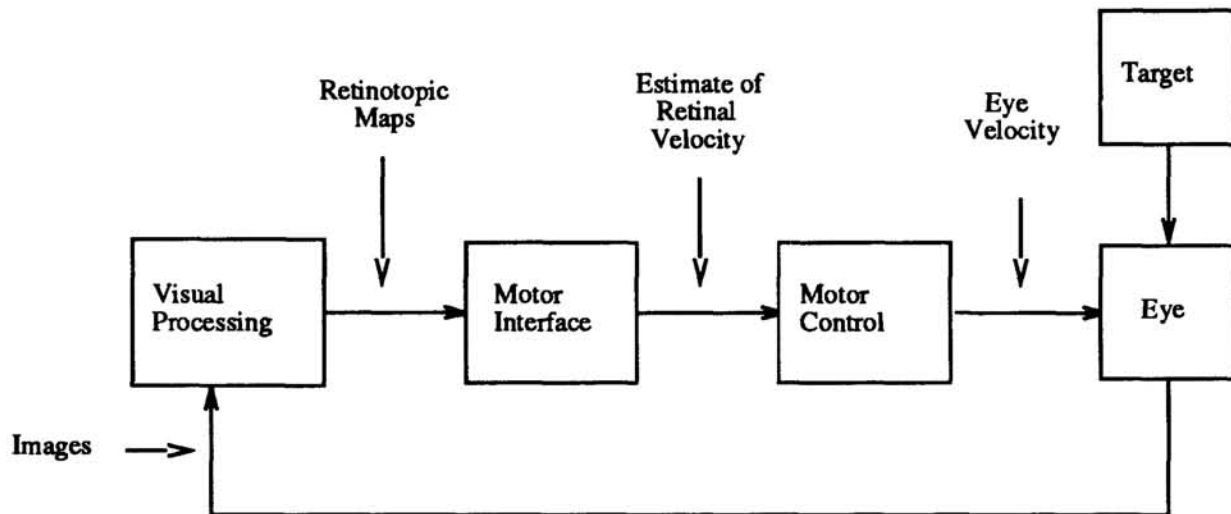

Figure 1: The overall structure of the visual tracking model.

# 1   Introduction

The fovea of the primate eye has a high density of photoreceptors. Images that fall within the fovea are perceived with high resolution. Perception of moving objects poses a particular problem for the visual system. If the eyes are fixed a moving image will be blurred. When the image moves out the of the fovea, resolution decreases. By moving their eyes to foveate and stabilize targets, primates ensure maximum perceptual resolution. In addition, active target tracking simplifies other tasks, such as spatial localization and spatial coordinate transformations (Ballard, 1991).

Visual tracking is a feedback process, in which the eyes are moved to stabilize and foveate the image of a target. Good visual tracking performance depends on accurate estimates of target velocity and a stable feedback controller. Although many visual tracking systems have been designed by engineers, the primate visual tracking system has yet to be matched in its ability to perform in complicated environments, with unrestricted targets, and over a wide variety of target trajectories. The study of the primate oculomotor system is an important step toward building a system that can attain primate levels of performance. The model presented here can accurately and stably track a variety of targets over a wide range of trajectories and is a first step toward achieving this goal.

Our model has four primary components: a model eye, a visual processing network, a motor interface network, and a motor control network (see Figure 1). The model eye receives a sequence of images from a changing visual world, synthetically rendered, and generates a time-varying output signal. The retinal signal is sent to the visual processing network which is similar in function to the motion processing areas of the visual cortex. The visual processing network constructs a distributed representation of image velocity. This representation is then used to estimate the velocity of the target on the retina. The retinal velocity of the target forms the input to the motor control network that drives the eye. The eye responds by rotating,

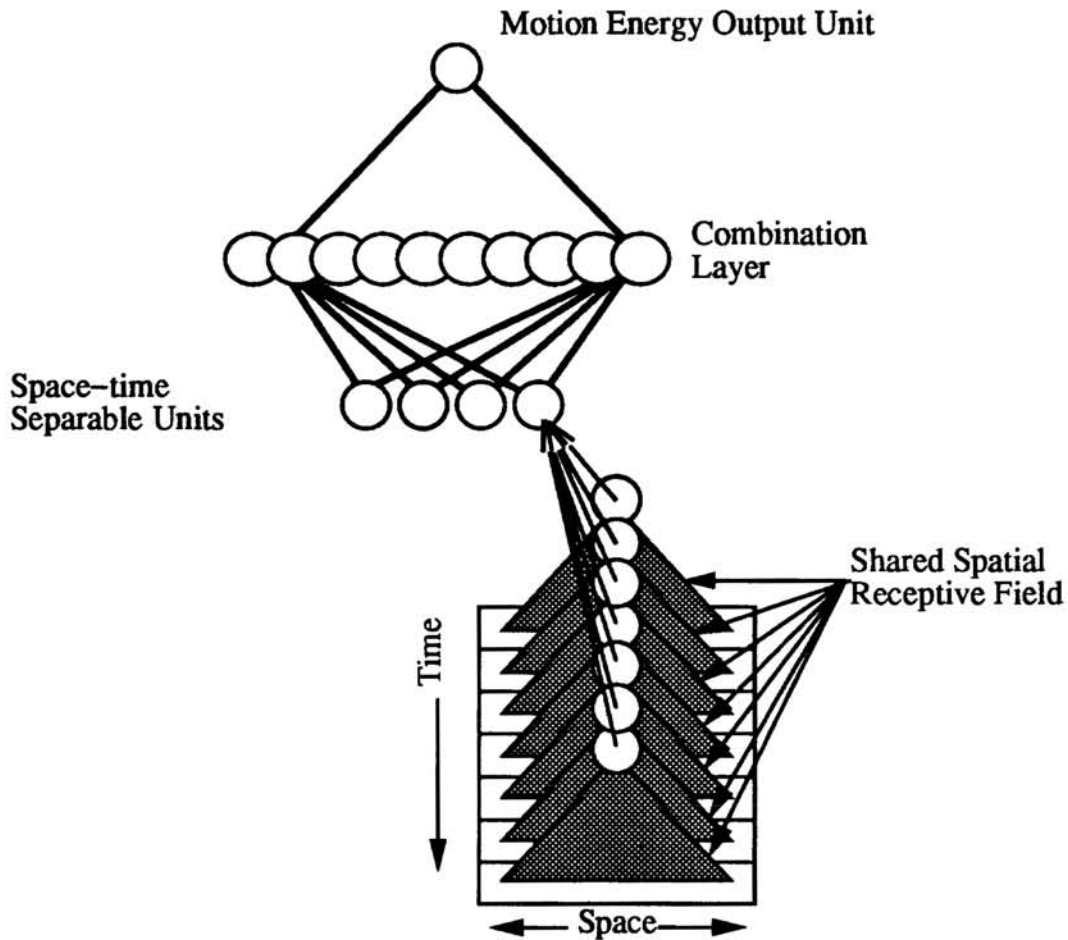

Figure 2: The structure of a motion energy unit. Each space-time separable unit has a receptive field that covers 16 pixels in space and 16 steps in time (for a total of 256 inputs). The shaded triangles denote complete projections.

which in turn affects incoming retinal signals.

If these networks function perfectly, eye velocity will match target velocity. Our model generates smooth eye motions to stabilize smoothly moving targets. It makes no attempt to foveate the image of a target. In primates, eye motions that foveate targets are called saccades. Saccadic mechanisms are largely separate from the smooth eye motion system (Lisberger et. al. 1987). We do not address them here.

In contrast with most engineered systems, our model is adaptive. The networks used in the model were trained using gradient descent[1]. This training process circumvented the need for a separate calibration of the visual tracking system.

## 2    Visual Processing

The middle temporal cortex (area MT) contains cells that are selective for the direction of visual motion. The neurons in MT are organized into a retinotopic map and small lesions in this area lead to selective impairment of visual tracking in the corresponding regions of the visual field (Newsome and Pare, 1988). The visual processing networks in our model contain directionally-selective processing units that are arranged in a retinotopic map. The spatio-temporal motion energy filter of Adelson and Bergen (Adelson and Bergen, 1985) has many of the properties of directionally-selective cortical neurons; it is used as the basis for our visual processing network. We constructed a four layer time-delay neural network that implements a motion energy calculation.

A single motion-energy unit can be constructed from four intermediate units having separable spatial and temporal filters. Adelson and Bergen demonstrate that two spatial filters (of even and odd symmetry) and two temporal filters (temporal derivatives for fast and slow speeds) are sufficient to detect motion. The filters are combined to construct 4 intermediate units which project to a single motion energy unit. Because the spatial and temporal properties of the receptive field are separable, they can be computed separately and convolved together to produce the final output. The temporal response is therefore the same throughout the extent of the spatial receptive field.

In our model, motion energy units are implemented as backpropagation networks. These units have a receptive field 16 pixels wide over a 16 time step window. Because the input weights are shared, only 32 parameters were needed for each space-time separable unit. Four space-time separable units project through a 16 unit combination layer to the output unit (see Figure 2). The entire network can be trained to approximate a variety of motion-energy filters.

We trained the motion energy network in two different ways: as a single multilayered network and in stages. Staged training proceded first by training intermediate units, then, with the intermediate units fixed, by training the three layer network that combines the intermediate units to produce a single motion energy output. The output unit is active when a pattern in the appropriate range of spatial frequencies moves through the receptive field with appropriate velocity. Many such units are required for a range of velocities, spatial frequencies, and spatial locations. We use six different types of motion energy units – each tuned to a different temporal frequency – at each of the central 48 positions of a 64 pixel linear retina. The 6 populations form a distributed, velocity-tuned representation of image motion for a total of 288 motion energy units.

In addition to the motion energy filters, static spatial frequency filters are also computed and used in the interface network, one for each band and each position for a total of 288 units.

We chose an adaptive network rather than a direct motion energy calculation because it allows us to model the dynamic nature of the visual signal with greater flexibility. However, this raises complications regarding the set of training images. Assuming 5 bits of information at each retinal position, there are well over 10 to the 100th possible input patterns. We explored sine waves, random spots and a variety of spatial pre-filters, and found low-pass filtered images of moving random spots worked best. Typically we began the training process from a plausible set of

weights, rather than from random values, to prevent the network from settling into an initial local minima. Training proceeded for days until good performance was obtained on a testing set.

Krauzlis and Lisberger (1989) have predicted that the visual stimulus to the visual tracking system in the brain contains information about the acceleration and impulse of the target as well as the velocity. Our motion energy networks are sensitive to target acceleration, producing transients for accelerating stimuli.

## 3   The Interface Network

The function of the interface is to take the distributed representation of the image motion and extract a single velocity estimate for the moving object. We use a relatively simple method that was adequate for tracking single objects without other moving distractors. The activity level of a single motion energy unit is ambiguous. First, it is necessary for the object to have a feature that is matched to the spatial frequency bandpass of the motion energy unit. Second, there is an array of units for each spatial frequency and the object will stimulate only a few of these at any given time. For instance, a large white object will have no features in its interior; a unit with its receptive field located in the interior can detect no motion. Conversely, detectors with receptive fields on the border between the object and the background will be strongly stimulated.

We use two stages of processing to extract a velocity. In the first stage, the motion energy in each spatial frequency band is estimated by summing the outputs of the motion energy filters across the retina weighted by the spatial frequency filter at each location. The six populations of spatial frequency units each yield one value. Next, a 6-6-1 feedforward network, trained using backpropagation, predicts target velocity from these values.

## 4   The Motor Control Network

In comparison with the visual processing network, the motor control network is quite small (see Figure 3). The goal of the network is to move the eye to stabilize the image of the object. The visual processing and interface networks convert images of the moving target into an estimate for the retinal velocity of the target. This retinal velocity can be considered a motor error. One approach to reducing this error is a simple proportional feedback controller, which drives the eye at a velocity proportional to the error. There is a large, 50-100 ms delay that occurs during visual processing in the primate visual system. In the presence of a large delay a proportional controller will either be inaccurate or unstable. For this reason simple proportional feedback is not sufficient to control tracking in the primate. Tracking can be made stable and accurate by including an internal positive feedback pathway to prevent instability while preserving accuracy (Robinson, 1971).

The motor control network was based on a model of the primate visual tracking motor control system by Lisberger and Sejnowski (1992). This recurrent artificial neural network includes both the smooth visual tracking system and the vestibulo-ocular system, which is important for compensating head movements. We use a

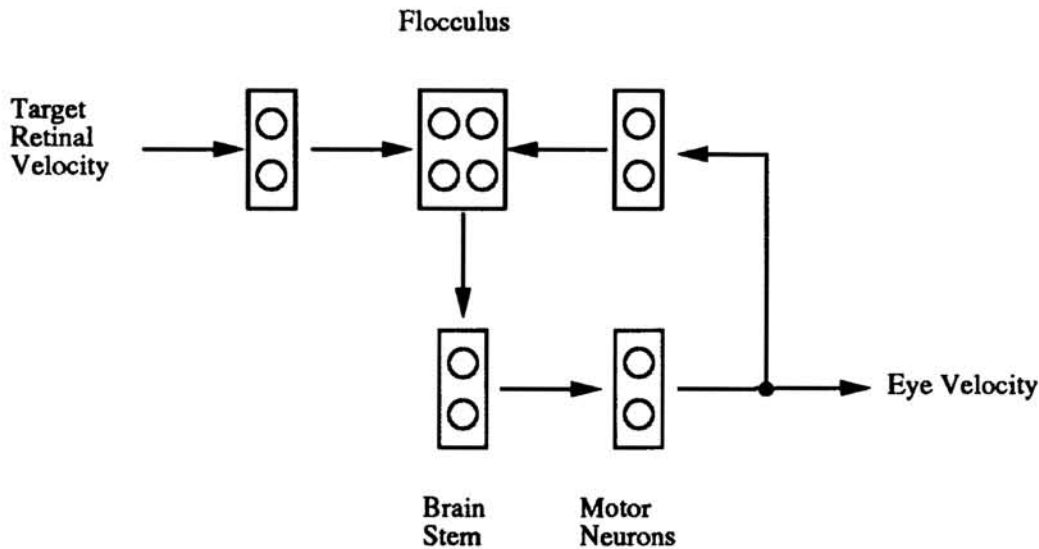

**Figure 3:** The structure of the recurrent network. Each circle is a unit. Units within a box are not interconnected and all units between boxes were fully interconnected as indicated by the arrows.

simpler version of that model that does not have vestibular inputs. The network is constructed from units with continuous smooth temporal responses. The state of a unit is a function of previous inputs and previous state:

$$s_j(t + \Delta t) = (1 - \tau \Delta t)s_j(t) + I\tau \Delta t$$

where $s_j(t)$ is the state of unit $j$ at time $t$, $\tau$ is a time constant and $I$ is the sigmoided sum of the weighted pre-synaptic activities. The resulting network is capable of smooth responses to inputs.

The motor control network has 12 units, each with a time constant of 5 ms (except for a few units with longer delay). There is a time delay of 50 ms between the interface network and control network. (see Figure 3). The input to the network is retinal target velocity, the output is eye velocity. The motor control network is trained to track a target in the presence of the visual delay.

The motor control network contains a positive feedback loop that is necessary to maintain accurate tracking even when the error signal falls to zero. The overall control network also contains a negative feedback loop since the output of the network affects subsequent inputs. The gradient descent optimization procedure uses the relationship between the output and the input during training—this relationship can be considered a model of the plant. It should be possible to use the same approach with more complex plants.

The control network was trained with the visual processing network frozen. A training example consists of an object trajectory and the goal trajectory for the eye. A standard recurrent network training paradigm is used to adjust the weights to minimize the error between actual outputs and desired outputs for step changes in target velocity.

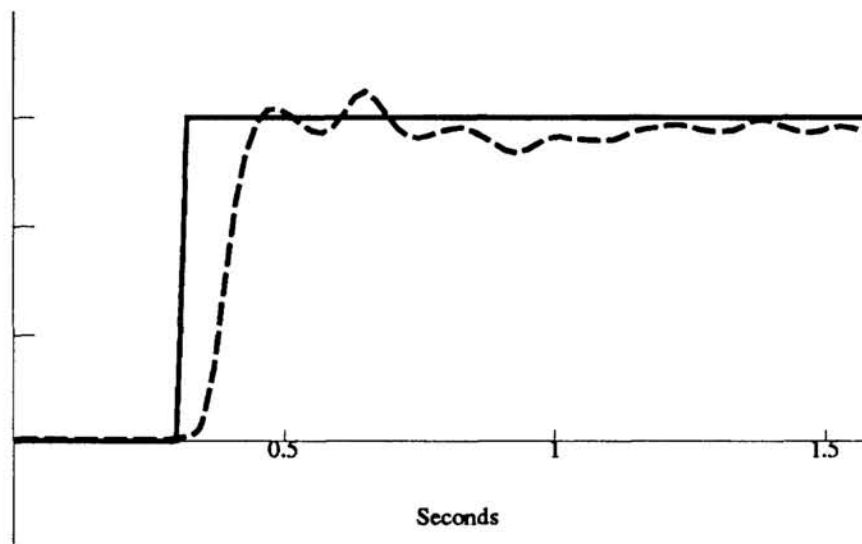

Seconds

Figure 4: Response of the eye to a step in target velocity of 30 degrees per second. The solid line is target velocity, the dashed line is eye velocity. This experiment was performed with a target that did not appear in the training set.

## 5   Performance

After training the network on a set of trajectories for a single target, the tracking performance was equally good on new targets. Tracking is accurate and stable - with little tendency to ring (see Figure 4). This good performance is surprising in the presence of a 50 millisecond delay in the visual feedback signal[2]. Stable tracking is not possible without the positive internal feedback loop in the model (eye velocity signal to the flocculus in Figure 3).

## 6   Limitations

The system that we have designed is a relatively small one having a one-dimensional retina only 64 pixels wide. The eye and the target can only move in one dimension—along the length of the retina. The visual analysis that is performed is not, however, limited to one dimension. Motion energy filters are easily generalized to a two-dimensional retina. Our approach should be extendable to the two-dimensional tracking problem.

The backgrounds of images that we used for tracking were featureless. The current system cannot distinguish target features from background features. Also, the interface network was designed to track a single object in the absence of moving distractors. The next step is to expand this interface to model the attentional phenomena observed in primate tracking, especially the process of initial target

acquisition.

## 7 Conclusion

In simulations, our eye tracking model performed well. Many additional difficulties must be addressed, but we feel this system can perform well under real-world real-time constraints. Previous work by Lisberger and Sejnowski (1992) demonstrates that this visual tracking model can be integrated with inertial eye stabilization—the vestibulo-ocular reflex. Ultimately, it should be possible to build a physical system using these design principles.

Every component of the system was designed using network learning techniques. The visual processing, for example, had a variety of components that were trained separately and in combinations. The architecture of the networks were based on the anatomy and physiology of the visual and oculomotor systems. This approach to reverse engineering is based on the existing knowledge of the flow of information through the relevant brain pathways.

It should also be possible to use the model to develop and test theories about the nature of biological visual tracking. This is just a first step toward developing a realistic model of the primate oculomotor system, but it has already provided useful predictions for the possible sites of plasticity during gain changes of the vestibulo-ocular reflex (Lisberger and Sejnowski, 1992).

## References

[1] E. H. Adelson and J. R. Bergen. Spatiotemporal energy models of the perception of motion. *Journal of the Optical Society of America*, 2(2):284–299, 1985.

[2] D. H. Ballard. Animate vision. *Artificial Intelligence*, 48:57–86, 1991.

[3] R.J. Krauzlis and S. G. Lisberger. A control systems model of smooth pursuit eye movements with realistic emergent properties. *Neural Computation*, 1:116–122, 1992.

[4] S. G. Lisberger, E. J. Morris, and L. Tychsen. *Ann. Rev. Neurosci.*, 10:97–129, 1987.

[5] S.G. Lisberger and T.J. Sejnowski. Computational analysis suggests a new hypothesis for motor learning in the vestibulo-ocular reflex. Submitted for publication., 1992.

[6] W.T. Newsome and E. B. Pare. A selective impairment of motion perception following lesions of the middle temporal visual area (MT). *J. Neuroscience*, 8:2201–2211, 1988.

[7] D. A. Robinson. Models of oculomotor neural organization. In P. Bach y Rita and C. C. Collins, editors, *The Control of Eye Movements*, page 519. Academic, New York, 1971.
